# Affine Independent Variational Inference

**Edward Challis**  **David Barber**
Department of Computer Science
University College London, UK
{edward.challis,david.barber}@cs.ucl.ac.uk

## Abstract

We consider inference in a broad class of non-conjugate probabilistic models based on minimising the Kullback-Leibler divergence between the given target density and an approximating 'variational' density. In particular, for generalised linear models we describe approximating densities formed from an affine transformation of independently distributed latent variables, this class including many well known densities as special cases. We show how all relevant quantities can be efficiently computed using the fast Fourier transform. This extends the known class of tractable variational approximations and enables the fitting for example of skew variational densities to the target density.

## 1 Introduction

Whilst Bayesian methods have played a significant role in machine learning and related areas (see [1] for an introduction), improving the class of distributions for which inference is either tractable or can be well approximated remains an ongoing challenge. Within this broad field of research, variational methods have played a key role by enabling mathematical guarantees on inferences (see [28] for an overview). Our contribution is to extend the class of approximating distributions beyond classical forms to approximations that can possess skewness or other non-Gaussian characteristics, while maintaining computational efficiency.

We consider approximating the normalisation constant $Z$ of a probability density function $p(\mathbf{w})$

$$p(\mathbf{w}) = \frac{1}{Z} \prod_{n=1}^{N} f_n(\mathbf{w}) \quad \text{with} \quad Z = \int \prod_{n=1}^{N} f_n(\mathbf{w}) d\mathbf{w} \tag{1.1}$$

where $\mathbf{w} \in \mathbb{R}^D$ and $f_n : \mathbb{R}^D \to \mathbb{R}^+$ are potential functions. Apart from special cases, evaluating $Z$ and other marginal quantities of $p(\mathbf{w})$ is difficult due to the assumed high dimensionality $D$ of the integral. To address this we may find an approximating density $q(\mathbf{w})$ to the target $p(\mathbf{w})$ by minimising the Kullback-Leibler (KL) divergence

$$\text{KL}(q(\mathbf{w})|p(\mathbf{w})) = \langle \log q(\mathbf{w}) \rangle_{q(\mathbf{w})} - \langle \log p(\mathbf{w}) \rangle_{q(\mathbf{w})} = -H\left[q(\mathbf{w})\right] - \langle \log p(\mathbf{w}) \rangle_{q(\mathbf{w})} \tag{1.2}$$

where $\langle f(x) \rangle_{p(x)}$ refers to taking the expectation of $f(x)$ with respect to the distribution $p(x)$ and $H\left[q(\mathbf{w})\right]$ is the differential entropy of the distribution $q(\mathbf{w})$. The non-negativity of the KL divergence provides the lower bound

$$\log Z \geq H\left[q(\mathbf{w})\right] + \sum_{n=1}^{N} \langle \log f_n(\mathbf{w}) \rangle := \mathcal{B}. \tag{1.3}$$

Finding the best parameters $\theta$ of the approximating density $q(\mathbf{w}|\theta)$ is then equivalent to maximising the lower bound on $\log Z$. This KL bounding method is constrained by the class of distributions

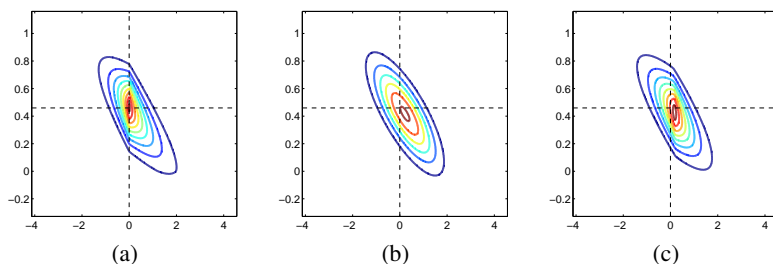

(a)                     (b)                     (c)

Figure 1: Two dimensional Bayesian sparse linear regression given a single data pair $\mathbf{x}, y$ using a Laplace prior $f_d(\mathbf{w}) \equiv \frac{1}{2\tau} e^{-|w_d|/\tau}$ with $\tau = 0.16$ and Gaussian likelihood $\mathcal{N}\left(y|\mathbf{w}^{\mathsf{T}}\mathbf{x}, \sigma_l^2\right)$, $\sigma_l^2 = 0.05$. (a) True posterior with $\log Z = -1.4026$. (b) Optimal Gaussian approximation with bound value $\mathcal{B}_G = -1.4399$. (c) Optimal AI generalised-normal approximation with bound value $\mathcal{B}_{AI} = -1.4026$.

$p(\mathbf{w})$ and $q(\mathbf{w})$ for which (1.3) can be efficiently evaluated. We therefore specialise on models of the form

$$p(\mathbf{w}) \propto \mathcal{N}\left(\mathbf{w}|\boldsymbol{\mu}, \boldsymbol{\Sigma}\right) \prod_{n=1}^{N} f_n(\mathbf{w}^{\mathsf{T}}\mathbf{x}_n) \qquad (1.4)$$

where $\{\mathbf{x}_n\}_{n=1}^{N}$ is a collection of fixed $D$ dimensional real vectors and $f_n : \mathbb{R} \to \mathbb{R}^+$; $\mathcal{N}\left(\mathbf{w}|\boldsymbol{\mu}, \boldsymbol{\Sigma}\right)$ denotes a multivariate Gaussian in $\mathbf{w}$ with mean $\boldsymbol{\mu}$ and covariance $\boldsymbol{\Sigma}$. This class includes Bayesian generalised linear models, latent linear models, independent components analysis and sparse linear models amongst others[1]. Many problems have posteriors that possess non-Gaussian characteristics resulting from strongly non-Gaussian priors or likelihood terms. For example, in financial risk modelling it is crucial that skewness and heavy tailed properties of the data are accurately captured [27]; similarly in inverse modelling, sparsity inducing priors can lead to highly non-Gaussian posteriors.

It is therefore important to extend the class of tractable approximating distributions beyond standard forms such as the multivariate Gaussian [20, 12, 2, 13]. Whilst mixtures of Gaussians [4, 10, 5] have previously been developed, these typically require additional bounds. Our interest here is to consider alternative multivariate distribution classes for which the KL method is more directly applicable[2].

## 2   Affine independent densities

We first consider independently distributed latent variables $\mathbf{v} \sim q_{\mathbf{v}}(\mathbf{v}|\boldsymbol{\theta}) = \prod_{d=1}^{D} q_{v_d}(v_d|\theta_d)$ with 'base' distributions $q_{v_d}$. To enrich the representation, we form the affine transformation $\mathbf{w} = \mathbf{A}\mathbf{v} + \mathbf{b}$ where $\mathbf{A} \in \mathbb{R}^{D \times D}$ is invertible and $\mathbf{b} \in \mathbb{R}^D$. The distribution on $\mathbf{w}$ is then[3]

$$q_{\mathbf{w}}(\mathbf{w}|\mathbf{A}, \mathbf{b}, \boldsymbol{\theta}) = \int \delta\left(\mathbf{w} - \mathbf{A}\mathbf{v} - \mathbf{b}\right) q_{\mathbf{v}}(\mathbf{v}|\boldsymbol{\theta}) d\mathbf{v} = \frac{1}{|\det\left(\mathbf{A}\right)|} \prod_d q_{v_d}\left(\left[\mathbf{A}^{-1}\left(\mathbf{w} - \mathbf{b}\right)\right]_d |\theta_d\right) \quad (2.1)$$

where $\delta\left(\mathbf{x}\right) = \prod_i \delta(x_i)$ is the Dirac delta function, $\boldsymbol{\theta} = [\theta_1, ..., \theta_d]$ and $[\mathbf{x}]_d$ refers to the $d^{th}$ element of the vector $\mathbf{x}$. Typically we assume the base distributions are homogeneous, $q_{v_d} \equiv q_v$. For instance, if we constrain each factor $q_{v_d}(v_d|\theta_d)$ to be the standard normal $\mathcal{N}\left(v_d|0, 1\right)$ then $q_{\mathbf{w}}(\mathbf{w}) = \mathcal{N}\left(\mathbf{w}|\mathbf{b}, \mathbf{A}\mathbf{A}^{\mathsf{T}}\right)$. By using, for example, Student's $t$, Laplace, logistic, generalised-normal or skew-normal base distributions, equation (2.1) parameterises multivariate extensions of these univariate distributions. This class of multivariate distributions has the important property that, unlike the

Gaussian, they can approximate skew and/or heavy-tailed $p(\mathbf{w})$. See figures 1, 2 and 3, for examples of two dimensional distributions $q_\mathbf{w}(\mathbf{w}|\mathbf{A}, \mathbf{b}, \boldsymbol{\theta})$ with skew-normal and generalised-normal base distributions used to approximate toy machine learning problems.

Provided we choose a base distribution class that includes the Gaussian as a special case (for example generalised-normal, skew-normal and asymptotically Student's $t$) we are guaranteed to perform at least as well as classical multivariate Gaussian KL approximate inference.

We note that we may arbitrarily permute the indices of $\mathbf{v}$. Furthermore, since every invertible matrix is expressible as $\mathbf{LUP}$ for $\mathbf{L}$ lower, $\mathbf{U}$ upper and $\mathbf{P}$ permutation matrices, without loss of generality, we may use an LU decomposition $\mathbf{A} = \mathbf{LU}$; this reduces the complexity of subsequent computations.

Whilst defining such Affine Independent (AI) distributions is straightforward, critically we require that the bound, equation (1.3), is fast to compute. As we explain below, this can be achieved using the Fourier transform both for the bound and its gradients. Full derivations, including formulae for skew-normal and generalised-normal base distributions, are given in the supplementary material.

## 2.1 Evaluating the KL bound

The KL bound can be readily decomposed as

$$\mathcal{B} = \underbrace{\log|\det(\mathbf{A})| + \sum_{d=1}^{D} H\left[q(v_d|\theta_d)\right]}_{\text{Entropy}} + \underbrace{\langle\log\mathcal{N}\left(\mathbf{w}|\boldsymbol{\mu}, \boldsymbol{\Sigma}\right)\rangle + \sum_{n=1}^{N}\left\langle\log f_n(\mathbf{w}^\mathsf{T}\mathbf{x}_n)\right\rangle}_{\text{Energy}} \quad (2.2)$$

where we used $H\left[q_\mathbf{w}(\mathbf{w})\right] = \log|\det(\mathbf{A})| + \sum_d H\left[q_{v_d}(v_d|\theta_d)\right]$ (see for example [8]). For many standard base distributions the entropy $H\left[q_{v_d}(v_d|\theta_d)\right]$ is closed form. When the entropy of a univariate base distribution is not analytically available, we assume it can be cheaply evaluated numerically. The energy contribution to the KL bound is the sum of the expectation of the log Gaussian term (which requires only first and second order moments) and the nonlinear 'site projections'. The non-linear site projections (and their gradients) can be evaluated using the methods described below.

### 2.1.1 Marginal site densities

Defining $y := \mathbf{w}^\mathsf{T}\mathbf{x}$, the expectation of the site projection for any function $g$ and fixed vector $\mathbf{x}$ is equivalent to a one-dimensional expectation, $\left\langle g\left(\mathbf{w}^\mathsf{T}\mathbf{x}\right)\right\rangle_{q_\mathbf{w}(\mathbf{w})} = \langle g(y)\rangle_{q_y(y)}$ with

$$q_y(y) = \int \delta(y - \mathbf{x}^\mathsf{T}\mathbf{w})q_\mathbf{w}(\mathbf{w})d\mathbf{w} = \int \delta(y - \boldsymbol{\alpha}^\mathsf{T}\mathbf{v} - \beta)q_\mathbf{v}(\mathbf{v})d\mathbf{v} \quad (2.3)$$

where $\mathbf{w} = \mathbf{Av} + \mathbf{b}$ and $\boldsymbol{\alpha} := \mathbf{A}^\mathsf{T}\mathbf{x}$, $\beta := \mathbf{b}^\mathsf{T}\mathbf{x}$. We can rewrite this $D$-dimensional integral as a one dimensional integral using the integral transform $\delta(x) = \int e^{2\pi itx}dt$:

$$q_y(y) = \int\int e^{2\pi it(y - \boldsymbol{\alpha}^\mathsf{T}\mathbf{v} - \beta)}\prod_{d=1}^{D} q_{v_d}(v_d)d\mathbf{v}dt = \int e^{2\pi i(t-\beta)y}\prod_{d=1}^{D}\tilde{q}_{u_d}(t)\,dt \quad (2.4)$$

where $\tilde{f}(t)$ denotes the Fourier transform of $f(x)$ and $q_{u_d}(u_d|\theta_d)$ is the density of the random variable $u_d := \alpha_d v_d$ so that $q_{u_d}(u_d|\theta_d) = \frac{1}{|\alpha_d|}q_{v_d}(\frac{u_d}{\alpha_d}|\theta_d)$. Equation(2.4) can be interpreted as the (shifted) inverse Fourier transform of the product of the Fourier transforms of $\{q_{u_d}(u_d|\theta_d)\}$.

Unfortunately, most distributions do not have Fourier transforms that admit compact analytic forms for the product $\prod_{d=1}^{D}\tilde{q}_{u_d}(t)$. The notable exception is the family of stable distributions for which linear combinations of random variables are also stable distributed (see [19] for an introduction). With the exception of the Gaussian (the only stable distribution with finite variance), Levy and Cauchy distributions, stable distributions do not have analytic forms for their density functions and are analytically expressible only in the Fourier domain. Nevertheless, when $q_\mathbf{v}(\mathbf{v})$ is stable distributed, marginal quantities of $\mathbf{w}$ such as $y$ can be computed analytically in the Fourier domain [3].

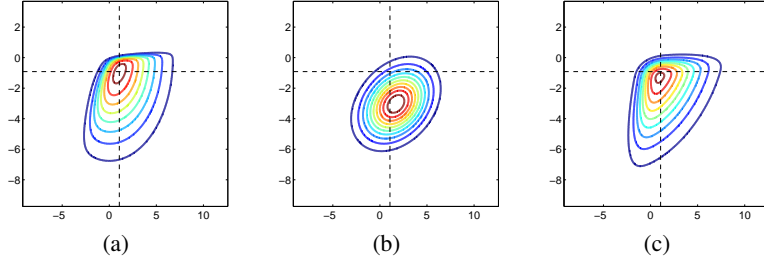

(a)　　　　　　　　　(b)　　　　　　　　　(c)

Figure 2: Two dimensional Bayesian logistic regression with Gaussian prior $\mathcal{N}\left(\mathbf{w}|\mathbf{0}, 10\mathbf{I}\right)$ and likelihood $f_n(\mathbf{w}) = \sigma(\tau_l c_n \mathbf{w}^\mathsf{T}\mathbf{x}_n)$, $\tau_l = 5$. Here $\sigma(x)$ is the logistic sigmoid and $c_n \in \{-1, +1\}$ the class labels; $N = 4$ data points. (a) True posterior with $\log Z = -1.13$. (b) Optimal Gaussian approximation with bound value $\mathcal{B}_G = -1.42$. (c) Optimal AI skew-normal approximation with bound value $\mathcal{B}_{AI} = -1.17$.

In general, therefore, we need to resort to numerical methods to compute $q_y(y)$ and expectations with respect to it. To achieve this we discretise the base distributions and, by choosing a sufficiently fine discretisation, limit the maximal error that can be incurred. As such, up to a specified accuracy, the KL bound may be exactly computed.

First we define the set of discrete approximations to $\{q_{u_d}(u_d|\theta_d)\}_{d=1}^D$ for $u_d := \alpha_d v_d$. These 'lattice' approximations are a weighted sum of $K$ delta functions

$$q_{u_d}(u_d|\theta_d) \approx \hat{q}_{u_d}(u_d) := \sum_{k=1}^K \pi_{dk}\delta\left(u_d - l_k\right) \quad \text{where} \quad \pi_{dk} = \int_{l_k - \frac{1}{2}\Delta}^{l_k + \frac{1}{2}\Delta} q(u_d|\theta_d)du_d. \quad (2.5)$$

The lattice points $\{l_k\}_{k=1}^K$ are spaced uniformly over the domain $[l_1, l_K]$ with $\Delta := l_{k+1} - l_k$. The weighting for each delta spike is the mass assigned to the distribution $q_{u_d}(u_d|\theta_d)$ over the interval $[l_k - \frac{1}{2}\Delta, l_k + \frac{1}{2}\Delta]$.

Given the lattice approximations to the densities $\{q_{u_d}(u_d|\theta_d)\}_{d=1}^D$ the fast Fourier transform (FFT) can be used to evaluate the convolution of the lattice distributions. Doing so we obtain the lattice approximation to the marginal $y = \mathbf{w}^\mathsf{T}\mathbf{x}$ such that (see supplementary section 2.2)

$$q_y(y) \approx \hat{q}_y(y) = \sum_{k=1}^K \delta(y - l_k - \beta)\rho_k \quad \text{where} \quad \boldsymbol{\rho} = \texttt{ifft}\left[\prod_{d=1}^D \texttt{fft}\left[\boldsymbol{\pi}_d'\right]\right]. \quad (2.6)$$

where $\boldsymbol{\pi}_d$ is padded with $(D-1)K$ zeros, $\boldsymbol{\pi}_d' := [\boldsymbol{\pi}_d, \mathbf{0}]$. The only approximation used in finding the marginal density is then the discretisation of the base distributions, with the remaining FFT calculations being exact. The time complexity for the above procedure scales $O\left(D^2 K \log KD\right)$.

### 2.1.2 Efficient site derivative computation

Whilst we have shown that the expectation of the site projections can be accurately computed using the FFT, how to cheaply evaluate the derivative of this term is less clear. The complication can be seen by inspecting the partial derivative of $\left\langle g(\mathbf{w}^\mathsf{T}\mathbf{x})\right\rangle$ with respect to $A_{mn}$

$$\frac{\partial}{\partial A_{mn}}\left\langle g(\mathbf{w}^\mathsf{T}\mathbf{x})\right\rangle_{q(\mathbf{w})} = x_n \int q_{\mathbf{v}}(\mathbf{v})g'\left(\mathbf{x}^\mathsf{T}\mathbf{A}\mathbf{v} + \mathbf{b}^\mathsf{T}\mathbf{x}\right)v_m d\mathbf{v}, \quad (2.7)$$

where $g'(y) = \frac{d}{dy}g(y)$. Naively, this can be readily reduced to a (relatively expensive) two dimensional integral. Critically, however, the computation can be simplified to a one dimensional integral. To see this we can write

$$\frac{\partial}{\partial A_{mn}}\left\langle g\left(\mathbf{w}^\mathsf{T}\mathbf{x}\right)\right\rangle = x_n \int g'(y)d_m(y)dy, \quad \text{where} \quad d_m(y) := \int v_m q_{\mathbf{v}}(\mathbf{v})\delta\left(y - \boldsymbol{\alpha}^\mathsf{T}\mathbf{v} - \beta\right)d\mathbf{v}.$$

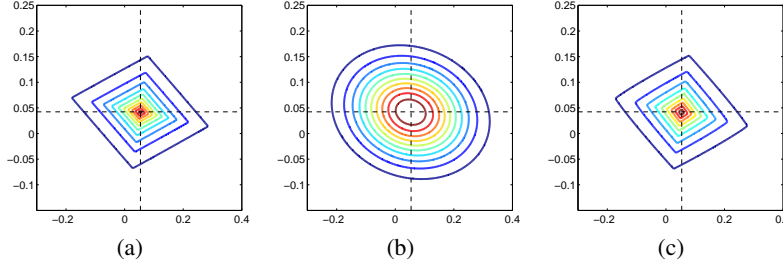

Figure 3: Two dimensional robust linear regression with Gaussian prior $\mathcal{N}(\mathbf{w}|\mathbf{0},\mathbf{I})$, Laplace likelihood $f_n(\mathbf{w}) = \frac{1}{2\tau_l}e^{-|y_n-\mathbf{w}^\mathsf{T}\mathbf{x}_n|/\tau_l}$ with $\tau_l = 0.1581$ and 2 data pairs $\mathbf{x}_n, y_n$. (a) True posterior with $\log Z = -3.5159$. (b) Optimal Gaussian approximation with bound value $\mathcal{B}_G = -3.6102$. (c) Optimal AI generalised-normal approximation with bound value $\mathcal{B}_{AI} = -3.5167$.

Here $d_m(y)$ is a univariate weighting function with Fourier transform:

$$\tilde{d}_m(t) = e^{-2\pi i t \beta}\tilde{e}_m(t)\prod_{d\neq m}\tilde{q}_{u_d}(t), \quad \text{where} \quad \tilde{e}_m(t) := \int \frac{u_m}{\alpha_m}q_{u_m}(u_m)e^{-2\pi i t u_m}du_m.$$

Since $\{\tilde{q}(t)\}_{d=1}^D$ are required to compute the expectation of $\langle g(\mathbf{w}^\mathsf{T}\mathbf{x})\rangle$ the only additional computations needed to evaluate all partial derivatives with respect to $\mathbf{A}$ are $\{\tilde{e}_d(t)\}_{d=1}^D$. Thus the complexity of computing the site derivative[4] is equivalent to the complexity of the site expectation of section 2.1.1.

Even for non-smooth functions $g$ the site gradient has the additional property that it is smooth, provided the base distributions are smooth. Indeed, this property extends to the KL bound itself, which has continuous partial derivatives, see supplementary material section 1. This means that gradient optimisation for AI based KL approximate inference can be applied, even when the target density is non-smooth. In contrast, other deterministic approximate inference routines are not universally applicable to non-smooth target densities – for instance the Laplace approximation and [10] both require the target to be smooth.

## 2.2 Optimising the KL bound

Given fixed base distributions, we can optimise the KL bound with respect to the parameters $\mathbf{A} = \mathbf{LU}$ and $\mathbf{b}$. Provided $\{f_n\}_{n=1}^N$ are log-concave the KL bound is jointly concave with respect to $\mathbf{b}$ and either $\mathbf{L}$ or $\mathbf{U}$. This follows from an application of the concavity result in [7] – see the supplementary material section 3.

Using a similar approach to that presented in section 2.1.2 we can also efficiently evaluate the gradients of the KL bound with respect to the parameters $\boldsymbol{\theta}$ that define the base distribution. These parameters $\boldsymbol{\theta}$ can control higher order moments of the approximating density $q(\mathbf{w})$ such as skewness and kurtosis. We can therefore jointly optimise over all parameters $\{\mathbf{A}, \mathbf{b}, \boldsymbol{\theta}\}$ simultaneously; this means that we can fully capitalise on the expressiveness of the AI distribution class, allowing us to capture non-Gaussian structure in $p(\mathbf{w})$.

In many modeling scenarios the best choice for $q_\mathbf{v}(\mathbf{v})$ will suggest itself naturally. For example, in section 4.1 we choose the skew-normal distribution to approximate Bayesian logistic regression posteriors. For heavy-tailed posteriors that arise for example in robust or sparse Bayesian linear regression models, one choice is to use the generalised-normal as base density, which includes the Laplace and Gaussian distributions as special cases. For other models, for instance mixed data factor analysis [15], different distributions for blocks of variables of $\{v_d\}_{d=1}^D$ may be optimal. However, in situations for which it is not clear how to select $q_\mathbf{v}(\mathbf{v})$, several different distributions can be assessed and then that which achieves the greatest lower bound $\mathcal{B}$ is preferred.

### 2.3 Numerical issues

The computational burden of the numerical marginalisation procedure described in section 2.1.1 depends on the number of lattice points used to evaluate the convolved density function $q_y(y)$. For the results presented we implemented a simple strategy for choosing the lattice points $[l_1, ..., l_K]$. Lattice end points were chosen[5] such that $[l_1, l_K] = [-6\sigma_y, 6\sigma_y]$ where $\sigma_y$ is the standard deviation of the random variable $y$: $\sigma_y^2 = \sum_d \alpha_d^2 \text{var}(v_d)$. From Chebyshev's inequality, taking six standard deviation end points guarantees that we capture at least 97% of the mass of $q_y(y)$. In practice this proportion is often much higher since $q_y(y)$ is often close to Gaussian for $D \gg 1$. We fix the number of lattice points used during optimisation to suit our computational budget. To compute the final bound value we apply the simple strategy of doubling the number of lattice points until the evaluated bound changes by less than $10^{-3}$ [6].

Fully characterising the overall accuracy of the approximation as a function of the number of lattice points is complex, see [24, 26] for a related discussion. One determining factor is the condition number (ratio of largest and smallest eigenvalues) of the posterior covariance. When the condition number is large many lattice points are needed to accurately discretise the set of distributions $\{q_{u_d}(u_d|\theta_d)\}_{d=1}^D$ which increases the time and memory requirements.

One possible route to circumventing these issues is to use base densities that have analytic Fourier transforms (such as a mixture of Gaussians). In such cases the discrete Fourier transform of $q_y(y)$ can be directly evaluated by computing the product of the Fourier transforms of each $\{q_{u_d}(u_d|\theta_d)\}_{d=1}^D$. The implementation and analysis of this procedure is left for future work.

The computational bottleneck for AI inference, assuming $N > D$, arises from computing the expectation and partial derivatives of the $N$ site projections. For parameters $\mathbf{w} \in \mathbb{R}^D$ this scales $O\left(ND^2 K \log DK\right)$. Whilst this might appear expensive it is worth considering it within the broader scope of lower bound inference methods. It was shown in [7] that exact Gaussian KL approximate inference has bound and gradient computations which scale $O\left(ND^2\right)$. Similarly, local variational bounding methods (see below) scale $O\left(ND^2\right)$ when implemented exactly.

## 3 Related methods

Another commonly applied technique to obtain a lower bound for densities of the form of equation (1.4) is the 'local' variational bounding procedure [14, 11, 22, 18]. Local bounding methods approximate the normalisation constant by bounding each non-conjugate term in the integrand, equation (1.1), with a form that renders the integral tractable. In [7] we showed that the Gaussian KL bound dominates the local bound in such models. Hence the AI KL method also dominates the local and Gaussian KL methods.

Other approaches increase the flexibility of the approximating distribution by expressing $q_{\mathbf{w}}(\mathbf{w})$ as a mixture. However, computing the entropy of a mixture distribution is in general difficult. Whilst one may bound the entropy term [10, 4], employing such additional bounds is undesirable since it limits the gains from using a mixture. Another recently proposed method to approximate integrals using mixtures is split mean field which iterates between constructing soft partitions of the integral domain and bounding those partitioned integrals [5]. The partitioned integrals are approximated using local or Gaussian KL bounds. Our AI method is complementary to the split mean field method since one may use the AI technique to bound each of the partitioned integrals and so achieve an improved bound.

## 4 Experiments

For the experiments below[6], AI KL bound optimisation is performed using L-BFGS[7]. Gaussian KL inference is implemented in all experiments using our own package[8].

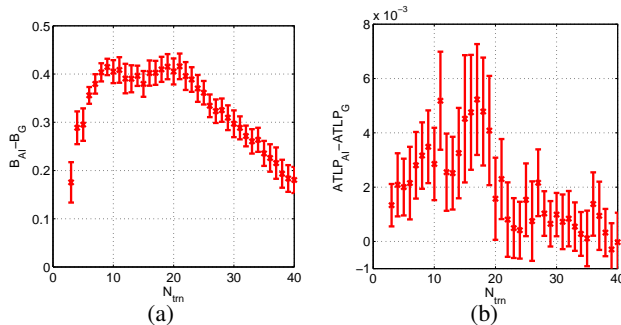

Figure 4: Gaussian KL and AI KL approximate inference comparison for a Bayesian logistic regression model with different training data set sizes $N_{trn}$. $\mathbf{w} \in \mathbb{R}^{10}$; Gaussian prior $\mathcal{N}(\mathbf{w}|\mathbf{0}, 5\mathbf{I})$; logistic sigmoid likelihood $f_n = \sigma(\tau_l c_n \mathbf{w}^\mathsf{T} \mathbf{x}_n)$ with $\tau_l = 5$; covariates $\mathbf{x}_n$ sampled from the standard normal, $\mathbf{w}^{true}$ sampled from the prior and class labels $c_n = \pm 1$ sampled from the likelihood. (a) Bound differences, $\mathcal{B}_{AI} - \mathcal{B}_G$, achieved using Gaussian KL and AI KL approximate inference for different training dataset sizes $N_{trn}$. Mean and standard errors are presented from 15 randomly generated models. A logarithmic difference of $0.4$ corresponds to $49\%$ improvement in the bound on the marginal likelihood. (b) Mean and standard error averaged test set log probability (ATLP) differences obtained with the Gaussian and AI approximate posteriors for different training dataset sizes $N_{trn}$. ATLP values calculated using $10^4$ test data points sampled from each model.

## 4.1 Toy problems

We compare the performance of Gaussian KL and AI KL approximate inference methods in three different two dimensional generalised linear models against the true posteriors and marginal likelihood values obtained numerically. See supplementary section 4 for derivations. Figure 1 presents results for a linear regression model with a sparse Laplace prior; the AI base density is chosen to be generalised-normal. Figure 2 demonstrates approximating a Bayesian logistic regression posterior, with the AI base distribution skew-normal. Figure 3 corresponds to a Bayesian linear regression model with the noise robust Laplace likelihood density and Gaussian prior; again the AI approximation uses the generalised-normal as the base distribution. The AI KL procedure achieves a consistently higher bound than the Gaussian case, with the AI bound nearly saturating at the true value of $\log Z$ in two of the models. In addition, the AI approximation captures significant non-Gaussian features of the posterior: the approximate densities are sparse in directions of sparsity of the posterior; their modes are approximately equal (where the Gaussian mode can differ significantly); tail behaviour is more accurately captured by the AI distribution than by the Gaussian.

## 4.2 Bayesian logistic regression

We compare Gaussian KL and AI KL approximate inference for a synthetic Bayesian logistic regression model. The AI density has skew-normal base distribution with $\theta_d$ parameterising the skewness of $v_d$. We optimised the AI KL bound jointly with respect to $\mathbf{L}, \mathbf{U}, \mathbf{b}$ and $\boldsymbol{\theta}$ simultaneously with convergence taking on average 8 seconds with $D = N = 10$, compared to $0.2$ seconds for Gaussian KL[9]. In figure 4 we plot the performance of the KL bound for the Gaussian versus the skew-normal AI approximation as we vary the number of datapoints. In (a) we plot the mean and standard error bound differences $\mathcal{B}_{AI} - \mathcal{B}_G$. For a small number of datapoints the bound difference is small. This difference increases up to $D = N$, and then decreases for larger datasets. This behaviour can be explained by the fact that when there are few datapoints the Gaussian prior dominates, with little difference therefore between the Gaussian and optimal AI approximation (which becomes effectively Gaussian). As more data is introduced, the non-Gaussian likelihood terms have a stronger impact and the posterior becomes significantly non-Gaussian. However as even more data is introduced the central limit theorem effect takes hold and the posterior becomes increasingly Gaussian. In (b) we

plot the mean and standard error differences for the averaged test set log probabilities (ATLP) calculated using the Gaussian and AI approximate posteriors obtained in each model. For each model and each training set size the ATLP is calculated using $10^4$ test points sampled from the model. The log test set probability of each test data pair $\mathbf{x}^*, c^*$ is calculated as $\log \langle p(c^*|\mathbf{w}, \mathbf{x}^*) \rangle_{q(\mathbf{w})}$ for $q(\mathbf{w})$ the approximate posterior. The bound differences can be seen to be strongly correlated with test set log probability differences, confirming that tighter bound values correspond to improved predictive performance.

### 4.3  Sparse robust kernel regression

In this experiment we consider sparse noise robust kernel regression. Sparsity is encoded using a Laplace prior on the weight vectors $\prod_d f_d(w_d)$ where $f_d(w_d) = e^{-|w_d|/\tau_p}/2\tau_p$. The Laplace distribution is also used as a noise robust likelihood $f_n(\mathbf{w}) = p(y_n|\mathbf{w}, \mathbf{k}_n) = e^{-|y_n - \mathbf{w}^\mathsf{T}\mathbf{k}_n|/\tau_l}/2\tau_l$ where $\mathbf{k}_n$ is the $n^{th}$ vector of the kernel matrix. The squared exponential kernel was used throughout with length scale $0.05$ and additive noise $1$, see [23]. In all experiments the prior and likelihood were fixed with $\tau_p = \tau_l = 0.16$. Three datasets are considered: Boston housing[10] ($D = 14$, $N_{trn} = 100$, $N_{tst} = 406$); Concrete Slump Test[11] ($D = 10$, $N_{trn} = 100$, $N_{tst} = 930$); a synthetic dataset constructed as described in [17] §5.6.1 ($D = 10$, $N_{trn} = 100$, $N_{tst} = 406$). Results are collected for each data set over 10 random training and test set partitions. All datasets are zero mean unit variance normalised based on the statistics of the training data.

| Dataset | $\bar{\mathcal{B}}_G$ | $\bar{\mathcal{B}}_{AI}$ | $\bar{\mathcal{B}}_{AI} - \bar{\mathcal{B}}_G$ | $ATLP_G$ | $ATLP_{AI}$ | $ATLP_{AI} - ATLP_G$ |
|---|---|---|---|---|---|---|
| Conc. CS. | $-2.08 \pm 0.09$ | $-2.06 \pm 0.09$ | $0.022 \pm 0.004$ | $-1.70 \pm 0.11$ | $-1.67 \pm 0.11$ | $0.024 \pm 0.010$ |
| Boston | $-1.28 \pm 0.05$ | $-1.25 \pm 0.05$ | $0.028 \pm 0.003$ | $-1.18 \pm 0.10$ | $-1.15 \pm 0.09$ | $0.023 \pm 0.006$ |
| Synthetic | $-2.49 \pm 0.10$ | $-2.46 \pm 0.10$ | $0.028 \pm 0.004$ | $-1.84 \pm 0.11$ | $-1.83 \pm 0.11$ | $0.009 \pm 0.009$ |

AI KL inference is performed with a generalised-normal base distribution. The parameters $\theta_d$ control the kurtosis of the base distributions $q(v_d|\theta_d)$; for simplicity we fix $\theta_d = 1.5$ and optimise jointly for $\mathbf{L}, \mathbf{U}, \mathbf{b}$. Bound optimisation took roughly 250 seconds for the AI KL procedure, compared to 5 seconds for the Gaussian KL procedure. Averaged results and standard errors are presented in the table above where $\bar{\mathcal{B}}$ denotes the bound value divided by the number of points in the training dataset. Whilst the improvements for these particular datasets are modest, the AI bound dominates the Gaussian bound in all three datasets, with predictive log probabilities also showing consistent improvement.

Whilst we have only presented experimental results for AI distributions with simple analytically expressible base distributions we note the method is applicable for any base distribution provided $\{q_{v_d}(v_d)\}_{d=1}^D$ are smooth. For example smooth univariate mixtures for $q_{v_d}(v_d)$ can be used.

## 5  Discussion

Affine independent KL approximate inference has several desirable properties compared to existing deterministic bounding methods. We've shown how it generalises on classical multivariate Gaussian KL approximations and our experiments confirm that the method is able to capture non-Gaussian effects in posteriors. Since we optimise the KL divergence over a larger class of approximating densities than the multivariate Gaussian, the lower bound to the normalisation constant is also improved. This is particularly useful for model selection purposes where the normalisation constant plays the role of the model likelihood.

There are several interesting areas open for further research. The numerical procedures presented in section 2.1 provide a general and computationally efficient means for inference in non-Gaussian densities whose application could be useful for a range of probabilistic models. However, our current understanding of the best approach to discretise the base densities is limited and further study of this is required, particularly for application in very large systems $D \gg 1$. It would also be useful to investigate using base densities that directly allow for efficient computation of the marginals $q_y(y)$ in the Fourier domain.

## Footnotes

[1]For $p(\mathbf{w})$ in this model class and Gaussian $q(\mathbf{w}) = \mathcal{N}\left(\mathbf{w}|\mathbf{m}, \mathbf{C}^{\mathsf{T}}\mathbf{C}\right)$, $\mathcal{B}$ is tighter than 'local' bounds [14, 11, 22, 18, 16]. For log-concave $f$, $\mathcal{B}$ is jointly concave in $(\mathbf{m}, \mathbf{C})$ for $\mathbf{C}$ the Cholesky matrix [7].

[2]The skew-normal $q(\mathbf{w})$ recently discussed in [21] possesses skew in one direction of parameter space only and is a special case of the AI skew-normal densities used in section 4.2.

[3]This construction is equivalent to a form of square noiseless Independent Components Analysis. See [9] and [25] for similar constructions.

[4]Further derivations and computational scaling properties are provided in supplementary section 2.

[5]For symmetric densities $\{q_{u_d}(u_d|\theta_d)\}$ we arranged that their mode coincides with the central lattice point.

[6]All experiments are performed in Matlab 2009b on a 32 bit Intel Core 2 Quad 2.5 GHz processor.

[7]We used the `minFunc` package (www.di.ens.fr/~mschmidt)

[8]`mloss.org/software/view/308/`

[9]We note that split mean field approximate inference was reported to take approximately 100 seconds for a similar logistic regression model achieving comparable results [20].

[10] archive.ics.uci.edu/ml/datasets/Housing

[11] archive.ics.uci.edu/ml/datasets/Concrete+Slump+Test

# References

[1] D. Barber. *Bayesian Reasoning and Machine Learning*. Cambridge University Press, 2012.

[2] D. Barber and C. M. Bishop. Ensemble Learning for Multi-Layer Networks. In *Advances in Neural Information Processing Systems, NIPS 10*, 1998.

[3] D. Bickson and C. Guestrin. Inference with Multivariate Heavy-Tails in Linear Models. In *Advances in Neural Information Processing Systems, NIPS 23*. 2010.

[4] C. M. Bishop, N. Lawrence, T. Jaakkola, and M. I. Jordan. Approximating Posterior Distributions in Belief Networks Using Mixtures. In *Advances in Neural Information Processing Systems, NIPS 10*, 1998.

[5] G. Bouchard and O. Zoeter. Split Variational Inference. In *International Conference on Artificial Intelligence and Statistics, AISTATS*, 2009.

[6] R. N. Bracewell. *The Fourier Transform and its Applications*. McGraw-Hill Book Co, Singapore, 2000.

[7] E. Challis and D. Barber. Concave Gaussian Variational Approximations for Inference in Large-Scale Bayesian Linear Models. In *International Conference on Artificial Intelligence and Statistics, AISTATS*, 2011.

[8] T. M. Cover and J. A. Thomas. *Elements of Information Theory*. Wiley, New York, 1991.

[9] J. T. A. S. Ferreira and M. F. J. Steel. A New Class of Skewed Multivariate Distributions with Applications To Regression Analysis. *Statistica Sinica*, 17:505–529, 2007.

[10] G. Gersman, M. Hoffman, and D. Blei. Nonparametric Variational Inference. In *International Conference on Machine Learning, ICML 29*, 2012.

[11] M. Girolami. A Variational Method for Learning Sparse and Overcomplete Representations. *Neural Computation*, 13:2517–2532, 2001.

[12] A. Graves. Practical Variational Inference for Neural Networks. In *Advances in Neural Information Processing Systems, NIPS 24*, 2011.

[13] A. Honkela and H. Valpola. Unsupervised Variational Bayesian Learning of Nonlinear Models. In *Advances in Neural Information Processing Systems, NIPS 17*, 2005.

[14] T. Jaakkola and M. Jordan. A Variational Approach to Bayesian Logistic Regression Problems and their Extensions. In *Artificial Intelligence and Statistics, AISTATS 6*, 1996.

[15] M. E. Khan, B. Marlin, G. Bouchard, and K. Murphy. Variational Bounds for Mixed-Data Factor Analysis. In *Advances in Neural Information Processing Systems, NIPS 23*, 2010.

[16] D. A. Knowles and T. Minka. Non-conjugate Variational Message Passing for Multinomial and Binary Regression. In *Advances in Neural Information Processing Systems, NIPS 23*. 2011.

[17] M. Kuss. *Gaussian Process Models for Robust Regression, Classification, and Reinforcement Learning*. PhD thesis, Technische Universität Darmstadt, Darmstadt, Germany, 2006.

[18] H. Nickisch and M. Seeger. Convex Variational Bayesian Inference for Large Scale Generalized Linear Models. In *International Conference on Machine Learning, ICML 26*, 2009.

[19] J. P. Nolan. *Stable Distributions - Models for Heavy Tailed Data*. Birkhauser, Boston, 2012. In progress, Chapter 1 online at academic2.american.edu/∼jpnolan.

[20] M. Opper and C. Archambeau. The Variational Gaussian Approximation Revisited. *Neural Computation*, 21(3):786–792, 2009.

[21] J. Ormerod. Skew-Normal Variational Approximations for Bayesian Inference. Technical Report CRG-TR-93-1, School of Mathematics and Statistics, University of Sydney, 2011.

[22] A. Palmer, D. Wipf, K. Kreutz-Delgado, and B. Rao. Variational EM Algorithms for Non-Gaussian Latent Variable Models. In *Advances in Neural Information Processing Systems, NIPS 20*, 2006.

[23] C. E. Rasmussen and C. K. I. Williams. *Gaussian Processes for Machine Learning*. MIT Press, 2006.

[24] P. Ruckdeschel and M. Kohl. General Purpose Convolution Algorithm in S4-Classes by means of FFT. Technical Report 1006.0764v2, arXiv.org, 2010.

[25] S. K. Sahu, D. K. Dey, and M. D. Branco. A New Class of Multivariate Skew Distributions with Applications to Bayesian Regression Models. *The Canadian Journal of Statistics / La Revue Canadienne de Statistique*, 31(2):129–150, 2003.

[26] P. Schaller and G. Temnov. Efficient and precise computation of convolutions: applying FFT to heavy tailed distributions. *Computational Methods in Applied Mathematics*, 8(2):187–200, 2008.

[27] C. Siddhartha, F. Nardari, and N. Shephard. Markov chain Monte Carlo methods for stochastic volatility models. *Journal of Econometrics*, 108(2):281–316, 2002.

[28] M. J. Wainwright and M. I. Jordan. Graphical Models, Exponential Families, and Variational Inference. *Foundations and Trends in Machine Learning*, 1(1-2):1–305, 2008.

